# A Probabilistic Algorithm Integrating Source Localization and Noise Suppression of MEG and EEG Data

**Johanna M. Zumer**
Biomagnetic Imaging Lab
Department of Radiology
Joint Graduate Group in Bioengineering
University of California, San Francisco
San Francisco, CA 94143-0628
johannaz@mrsc.ucsf.edu

**Hagai T. Attias**
Golden Metallic, Inc.
San Francisco, CA
htattias@goldenmetallic.com

**Kensuke Sekihara**
Dept. of Systems Design and Engineering
Tokyo Metropolitan, University
Tokyo, 191-0065 Japan
ksekiha@cc.tmit.ac.jp

**Srikantan S. Nagarajan**
Biomagnetic Imaging Lab
Department of Radiology
Joint Graduate Group in Bioengineering
University of California, San Francisco
San Francisco, CA 94143-0628
sri@mrsc.ucsf.edu

## Abstract

We have developed a novel algorithm for integrating source localization and noise suppression based on a probabilistic graphical model of stimulus-evoked MEG/EEG data. Our algorithm localizes multiple dipoles while suppressing noise sources with the computational complexity equivalent to a single dipole scan, and is therefore more efficient than traditional multidipole fitting procedures. In simulation, the algorithm can accurately localize and estimate the time course of several simultaneously-active dipoles, with rotating or fixed orientation, at noise levels typical for averaged MEG data. Furthermore, the algorithm is superior to beamforming techniques, which we show to be an approximation to our graphical model, in estimation of temporally correlated sources. Success of this algorithm for localizing auditory cortex in a tumor patient and for localizing an epileptic spike source are also demonstrated.

## 1 Introduction

Mapping functional brain activity is an important problem in basic neuroscience research as well as clinical use. Clinically, such brain mapping procedures are useful to guide neurosurgical planning, navigation, and tumor and epileptic spike removal, as well as guiding the surgeon as to which areas of the brain are still relevant for cognitive and motor function in each patient.

Many non-invasive techniques have emerged for functional brain mapping, such as functional magnetic resonance imaging (fMRI) and electromagnetic source imaging (ESI). Although fMRI is the most popular method for functional brain imaging with high spatial resolution, it suffers from poor temporal resolution since it measures blood oxygenation level signals with fluctuations in the order of seconds. However, dynamic neuronal activity has fluctuations in the sub-millisecond time-scale that can only be directly measured with electromagnetic source imaging (ESI). ESI refers to imaging of neuronal activity using magnetoencephalography (MEG) and electroencephalography (EEG)

data. MEG refers to measurement of tiny magnetic fields surrounding the head and EEG refers to measurement of voltage potentials using an electrode array placed on the scalp.

The past decade has shown rapid development of whole-head MEG/EEG sensor arrays and of algorithms for reconstruction of brain source activity from MEG and EEG data. Source localization algorithms, which can be broadly classified as parametric or tomographic, make assumptions to overcome the ill-posed inverse problem. Parametric methods, including equivalent current dipole (ECD) fitting techniques, assume knowledge about the number of sources and their approximate locations. A single dipolar source can be localized well, but ECD techniques poorly describe multiple sources or sources with large spatial extent. Alternatively, tomographic methods reconstruct an estimate of source activity at every grid point across the whole brain. Of many tomographic algorithms, the adaptive beamformer has been shown to have the best spatial resolution and zero localization bias [1, 2].

All existing methods for brain source localization are hampered by the many types of noise present in MEG/EEG data. The magnitude of the stimulus-evoked neural sources are on the order of noise on a single trial, and so typically 50-200 trials are needed to average in order to distinguish the sources above noise. This can be time-consuming and difficult for a subject or patient to hold still or pay attention through the duration of the experiment. Gaussian thermal noise is present at the sensors themselves. Background room interference such as from powerlines and electronic equipment can be problematic. Biological noise such as heartbeat, eyeblink or other muscle artifact can also be present. Ongoing brain activity itself, including the drowsy-state alpha ($\sim$10Hz) rhythm can drown out evoked brain sources. Finally, most localization algorithms have difficulty in separating neural sources of interest that have temporally overlapping activity.

Noise in MEG and EEG data is typically reduced by a variety of preprocessing algorithms before being used by source localization algorithms. Simple forms of preprocessing include filtering out frequency bands not containing a brain signal of interest. Additionally and more recently, ICA algorithms have been used to remove artefactual components, such as eyeblinks. More sophisticated techniques have also recently been developed using graphical models for preprocessing prior to source localization [3, 4].

This paper presents a probabilistic modeling framework for MEG/EEG source localization that is robust to interference and noise. The framework uses a probabilistic hidden variable model that describes the observed sensor data in terms of activity from unobserved brain and interference sources. The unobserved source activities and model parameters are inferred from the data by a Variational-Bayes Expectation-Maximization algorithm. The algorithm then creates a spatiotemporal image of brain activity by scanning the brain, inferring the model parameters and variables from sensor data, and using them to compute the likelihood of a dipole at each grid location in the brain. We also show that an established source localization method, the minimum variance adaptive beamformer (MVAB), is an approximation of our framework.

## 2  Probabilistic model integrating source localization and noise suppression

This section describes the generative model for the data. We assume that the MEG/EEG data has been collected such that stimulus onset or some other experimental marker indicated the 'zero' time point. Ongoing brain activity, biological noise, background environmental noise, and sensor noise are present in both pre-stimulus and post-stimulus periods; however, the evoked neural sources of interest are only present in the post-stimulus time period. We therefore assume that the sensor data can be described as coming from four types of sources: (1) evoked source at a particular voxel (grid point), (2) all other evoked sources not at that voxel, (3) all background noise sources with spatial covariance at the sensors (including brain, biological, or environmental sources), and (4) sensor noise. We first infer the model describing source types (3) and (4) from the pre-stimulus data, then fix certain quantities (described in section 2.2) and infer the full model describing the remaining source types (1) and (2) from the post-stimulus data (described in section 2.1). After inference of the model, a map of the source activity is created as well as a map of the likelihood of activity across voxels.

Let $y_n$ denote the $K \times 1$ vector of sensor data for time point $n$, where $K$ is the number of sensors (typically 200). Time ranges from $-N_{pre} : 0 : N_{post} - 1$ where $N_{pre}$ ($N_{post}$) indicates the number

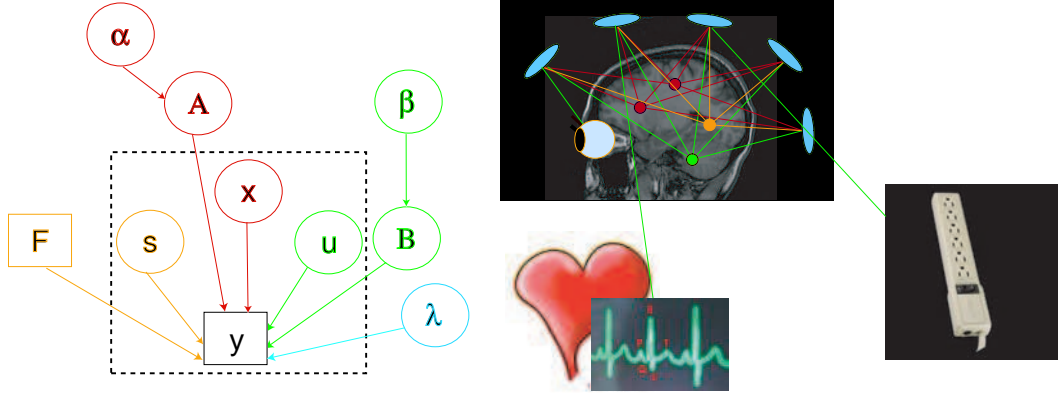

Figure 1: (Left) Graphical model for proposed algorithm. Variables are inside dotted box, parameters outside dotted box. Values in circles unknown and learned from the model, and values in squares known. (Right) Representation of factors influencing the data recorded at the sensors. In orange, a post-stimulus source at the voxel of interest, focused on by the lead field F. In red, other post-stimulus sources not at that particular voxel. In green, all background sources, including ongoing brain activity, eyeblinks, heartbeat, and electrical noise. In blue, thermal noise present in each sensor.

of time samples in the pre-(post-)stimulus period. The generative model for data $y_n$ is

$$y_n = \begin{cases} Bu_n + v_n & n = -N_{pre}, \dots, -1 \\ F^r s_n^r + A^r x_n^r + Bu_n + v_n & n = 0, \dots, N_{post} - 1 \end{cases} \quad (1)$$

The $K \times 3$ forward lead field matrix $F^r$ represents the physical (and linear) relationship between a dipole source at voxel $r$ for each orientation, and its influence on sensor $k = 1 : K$ [5]. The lead field $F^r$ is calculated from knowing the geometry of the source location to the sensor location, as well as the conducting medium in which the source lies: the human head is most commonly approximated as a single-shell sphere volume conductor. The source activity $s_n^r$ is a $3 \times 1$ vector of dipole strength in each of the three orientations at time $n$ for the voxel $r$. The $K \times L$ matrix $A$ and the $L \times 1$ vector $x_n$ represent the post-stimulus mixing matrix and evoked non-localized factors, respectively, corresponding to source type (2) discussed above. The $K \times M$ matrix $B$ and the $M \times 1$ vector $u_n$ represent the background mixing matrix and background factors, respectively. The $K \times 1$ vector $v_n$ represents the sensor-level noise. All quantities depend on $r$ in the post-stimulus period except for $B, u_n$ and $\lambda$ (the sensor precision), which will be learned from the pre-stimulus data and fixed as the other quantities are learned for each voxel. Note however the posterior update for $\bar{u}_n$ does depend on the voxel $r$. The graphical model is shown in Fig. 1. This generative model becomes a probabilistic model when we specify prior distributions, as described in the next two sections.

## 2.1 Localization of evoked sources learned from post-stimulus data

In the stimulus-evoked paradigm, the source strength at each voxel is learned from the post-stimulus data. The background mixing matrix $B$ and sensor noise precision $\lambda$ are fixed, after having been learned from the pre-stimulus data, described in section 2.2. We assume those quantities remain constant through the post-stimulus period and are independent of source location. We assume Gaussian prior distributions on the source factors and interference factors. We further make the assumption that the signals are independent and identically distributed (i.i.d.) across time. The source factors have prior precision given by the $3 \times 3$ matrix $\Phi$, which relates to the strength of the dipole in each of 3 orientations. All Normal distributions specified in this paper are defined by their mean and precision (inverse covariance).

$$p(s) = \prod_n p(s_n); \quad p(s_n) = \prod_{j=1}^{3} p(s_{jn}) = \mathcal{N}(0, \Phi) \quad (2)$$

The interference and background factors are assumed to have identity precision. To complete specification of this model, we need to specify prior distributions on the model parameters. We use a

conjugate prior for the interference mixing matrix $A$, where the $\alpha_j$ is a hyperparameter over the $j$th column of $A$ and $\lambda_i$ is the precision of the $i$th sensor. The hyperparameter $\alpha$ (a diagonal matrix) provides a robust mechanism for automatic model order selection, so that the optimal size of $A$ is inferred from the data through $\alpha$.

$$p(x) = \prod_n p(x_n); \quad p(x_n) = \prod_{j=1}^{L} p(x_{jn}) = \mathcal{N}(0, I);$$

$$p(u) = \prod_n p(u_n); \quad p(u_n) = \prod_{j=1}^{M} p(u_{jn}) = \mathcal{N}(0, I)$$

$$p(A) = \prod_{ij} \mathcal{N}(A_{ij}|0, \lambda_i \alpha_j) \tag{3}$$

We now specify the full model:

$$p(y|s, x, u, A, B) = \prod_n p(y_n|s_n, x_n, u_n, A, B);$$

$$p(y_n|s_n, x_n, u_n, A, B, \lambda) = \mathcal{N}(y_n|Fs_n + Ax_n + Bu_n, \lambda) \tag{4}$$

Exact inference on this model is intractable using the joint posterior over the interference factors and interference mixing matrix; thus the following variational-Bayesian approximation for the posteriors is used:

$$p(s, x, A|y) \approx q(s, x, A|y) = q(s, x|y)q(A|y) \tag{5}$$

We learn the hidden variables and parameters from the post-stimulus data, iterating through each voxel in the brain, using a variational-Bayesian Expectation-Maximization (EM) algorithm. All variables, parameters and hyperparameters are hidden and are learned from the data. In place of maximizing the $\log p(y)$, which would be mathematically intractable, we maximize a lower bound to $\log p(y)$ defined by $\mathcal{F}$ in the following equation

$$\mathcal{F} = \int dx\ ds\ dA\ q(s, x, A|y)\ [\log p(y, s, x, A) - \log q(s, x, A|y)]$$

$$= \log\ p(y) - \mathrm{KL}[q(s, x, A|y)||p(s, x, A|y)] \tag{6}$$

where $KL(q||p)$ is the Kullback-Leibler divergence between distributions q and p. $\mathcal{F}$ is equal to $\log p(y)$ when the approximation in Eq. 5 is true, thus making the $KL$-distance zero. We use a variational-Bayesian EM algorithm which alternately maximizes the function $\mathcal{F}$ with respect to the posteriors $q(s, x|y)$ and $q(A|y)$. In the E-step, $\mathcal{F}$ is maximized w.r.t. $q(s, x|y)$, keeping $q(A|y)$ constant, and the sufficient statistics of the hidden variables are computed. In the M-step, $\mathcal{F}$ is maximized w.r.t. $q(A|y)$, keeping $q(s, x|y)$ constant, and the MAP estimate of the parameters and hyperparameters are computed. In the E-step, the posterior distribution of the background factors given the data is computed:

$$q(x'_n|y_n) = \mathcal{N}(\bar{x}'_n, \Gamma);$$

$$\bar{x}'_n = \Gamma^{-1}\bar{A}'^T \lambda y_n; \qquad \Gamma = \bar{A}'^T \lambda \bar{A}' + K\Psi + I' \tag{7}$$

where we define:

$$\bar{x}'_n = \begin{pmatrix} \bar{s}_n \\ \bar{x}_n \\ \bar{u}_n \end{pmatrix}; \quad \bar{A}' = \begin{pmatrix} F & \bar{A} & \bar{B} \end{pmatrix}; \quad I' = \begin{pmatrix} \Phi & 0 & 0 \\ 0 & I & 0 \\ 0 & 0 & I \end{pmatrix}; \quad \Psi = \begin{pmatrix} 0 & 0 & 0 \\ 0 & \Psi_{AA} & 0 \\ 0 & 0 & 0 \end{pmatrix} \tag{8}$$

In the M-step, we maximize the function $\mathcal{F}$ w.r.t. $q(A|y)$ holding $q(s, x|y)$ fixed. We update the posterior distribution of the interference mixing matrix $A$ including its precision $\Psi_{AA}$. Note that the lead field $F$ is fixed based on the geometry of the sensors relative to the head, and $\bar{B}$ was learned and fixed from the pre-stimulus data. The sensor noise precision $\lambda$ is also kept fixed from the pre-stimulus period. The MAP values of the hyperparameter $\alpha$ and source factor precision $\Phi$ are learned here from the post-stimulus data.

$$\bar{A} = (R_{yx} - FR_{sx} - \bar{B}R_{ux})\Psi_{AA}; \quad \Psi_{AA} = (R_{xx} + \alpha)^{-1};$$

$$\Phi^{-1} = \frac{1}{N}R_{ss}; \quad \alpha^{-1} = \mathrm{diag}(\frac{1}{K}\bar{A}^T \lambda \bar{A} + \Psi_{AA}) \tag{9}$$

The matrices, such as $R_{yx}$, represent the posterior covariance between the two subscripts and explicit definitions are omitted for space. In each iteration of EM, the marginal likelihood is increased. The variational likelihood function (the lower bound on the exact marginal likelihood) is given as follows:

$$\mathcal{L}^r = \frac{N}{2}\log\frac{|\lambda||\Phi^r|}{|\Gamma^r|} - \frac{1}{2}\sum_{n=1}^{N}\left(y_n^T\lambda y_n - \bar{x}_n'^{Tr}\Gamma^r\bar{x}_n'^{r}\right) + \frac{K}{2}\log|\alpha^r||\Psi^r| \tag{10}$$

This likelihood function is dependent on the source voxel $r$ and thus a map of the likelihood across the brain can be displayed. Furthermore, we can also plot an image of the source power estimates and the time course of activity at each voxel.

We note that the computational complexity of the proposed algorithm is on the order $O(KLNS)$, roughly equivalent to a single dipole scan, which is of order $O(N(K^2+S))$. These are much smaller than the complexity of a multi-dipole scan which is order $O(NS^P)$ where P is the number of dipoles, and if S represents roughly several thousand voxels. We further note that the number of hidden variables to be estimated is less than the number of data points observed, thus not posing significant problems for estimation accuracy.

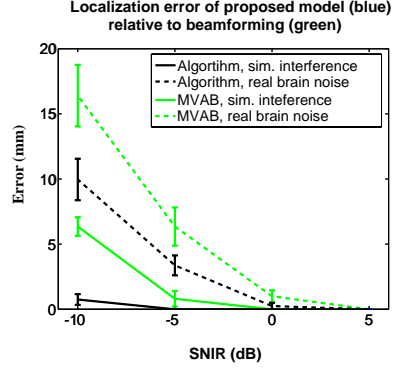

Figure 2: Performance of algorithm relative to beamforming for simulated datasets. See text for details.

## 2.2 Separation
### of background sources learned from pre-stimulus data

We learn the background mixing matrix and sensor noise precision from the pre-stimulus data using a variational-Bayes factor analysis model. We assume Gaussian prior distributions on the background factors and sensor noise, with zero mean and identity precision; we assume a flat prior on the sensor precision. We again use a conjugate prior for the background mixing matrix $B$, where $\beta_j$ is a hyperparameter, similar to the expression for the interference mixing matrix. All variables, parameters and hyperparameters are hidden and are learned from the pre-stimulus data. We make the variational-Bayesian approximation for the background mixing matrix and background factors $p(u, B|y) \approx q(u, B|y) = q(u|y)q(B|y)$. In the E-step, we maximize the function $\mathcal{F}$ w.r.t. $q(u|y)$ holding $q(B|y)$ fixed. We update the posterior distribution of the factors:

$$q(u|y) = \prod_n q(u_n|y_n); q(u_n|y_n) = \mathcal{N}(\bar{u}_n, \gamma)$$

$$\bar{u}_n = \gamma^{-1}B^T\lambda y_n; \gamma = \bar{B}^T\lambda\bar{B} + K\psi^{-1} + I$$

In the M-step, we compute the full posterior distribution of the background mixing matrix $B$, including its precision matrix $\psi$, and the MAP estimates of the noise precision $\lambda$ and the hyperparameter $\beta$. We assume the noise precision is diagonal.

$$\bar{B} = R_{yu}\psi; \quad \psi = (R_{uu} + \beta)^{-1}$$

$$\beta^{-1} = \text{diag}(\frac{1}{K}\bar{B}^T\lambda\bar{B} + \psi); \quad \lambda^{-1} = \frac{1}{N}\text{diag}(R_{yy} - \bar{B}R_{yu}^T) \tag{11}$$

## 2.3 Relationship to minimum-variance adaptive beamforming

Minimum variance adaptive beamforming (MVAB) is one of the best performing source localization techniques. MVAB estimates the dipole source time series by $\hat{s}_n = W_{MVAB}y_n$, where $W_{MVAB} = (F^T R_{yy}^{-1} F)^{-1}F^T R_{yy}^{-1}$ and $R_{yy}$ is the measured data covariance matrix. Thus, MVAB also has computational complexity equivalent to a single-dipole scan, on the order $O(K^2 + S)$. MVAB attempts to suppress interference, but recent studies have shown the MVAB is ineffective in cancellation of interference from other brain sources, especially if there are many such sources. In this section, we derive that MVAB is an approximation to inference on our model.

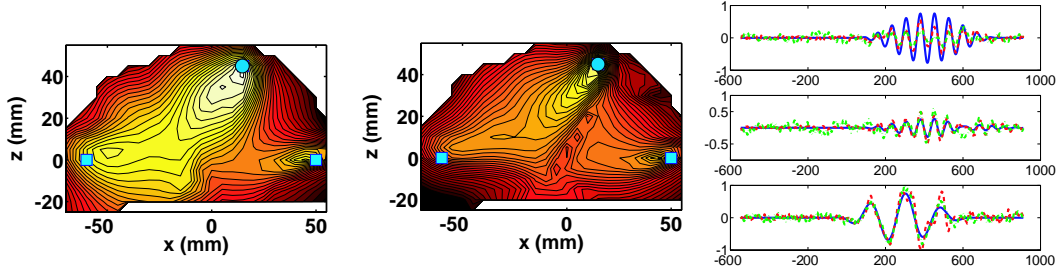

Figure 3: Example of algorithm and MVAB for correlated source simulation. See text for details

We start by rewriting Eq. (1) as $y_n = F s_n + z_n$, where $z_n$ is termed the total noise and is given by $z_n = A x_n + B u_n + v_n$. It has mean zero and precision matrix $\Upsilon = (AA^T + BB^T + \lambda^{-1})^{-1}$. Assuming we have estimated the model parameters $A, B, \lambda, \Phi$, the MAP estimate of the dipole source time series is $\bar{s}_n = W y_n$, where $W = \Gamma^{-1} F^T \Upsilon$ and $\Gamma = F^T \Upsilon F + \Phi$. It can be shown that this expression is equivalent to Eq. 8.

In the infinite data limit, the data covariance satisfies $R_{yy} = F \Phi^{-1} F^T + \Upsilon^{-1}$. Its inverse is found, using the matrix inversion lemma, to be $R_{yy}^{-1} = \Upsilon - \Upsilon F \Gamma^{-1} F^T \Upsilon$. Hence, we obtain

$$F^T R_{yy}^{-1} = (I - F^T \Upsilon F \Gamma^{-1}) F^T \Upsilon = \Phi \Gamma^{-1} F^T \Upsilon \qquad (12)$$

where the last step used the expression for $\Gamma$. Next, we approximate $\Gamma \approx F^T \Upsilon F$. We then use Eq. (12) to obtain:

$$W \approx (F^T \Upsilon F)^{-1} F^T \Upsilon = (F^T \Upsilon F)^{-1} \Gamma \Phi^{-1} \Phi \Gamma^{-1} F^T \Upsilon = (F^T R_{yy}^{-1} F)^{-1} F^T R_{yy}^{-1} = W_{MVAB}$$

## 3   Results

### 3.1   Simulations

The proposed method was tested in a variety of realistic source configurations reconstructed on a 5mm voxel grid. A single-shell spherical volume conductor model was used to calculate the forward lead field [5]. Simulated datasets were constructed by placing Gaussian-damped sinusoidal time courses at specific locations inside a voxel grid based on realistic head geometry. Sources were assumed to be present in the post-stimulus period with 437 samples along with a pre-stimulus period of 263 samples.

In the "noise-alone (NA)" cases, Gaussian noise only was added to all time points at the sensors. In the "interference (IN)" cases, Gaussian noise time courses occurring in both pre- and post-stimulus periods, representing simulated "ongoing" activity, were placed at 50 random locations throughout the brain voxel grid, and their activity was projected onto the sensors and added to both the sensor noise and source activity. Finally, in the "real (RE)" cases, 700 samples of real MEG sensor data averaged over

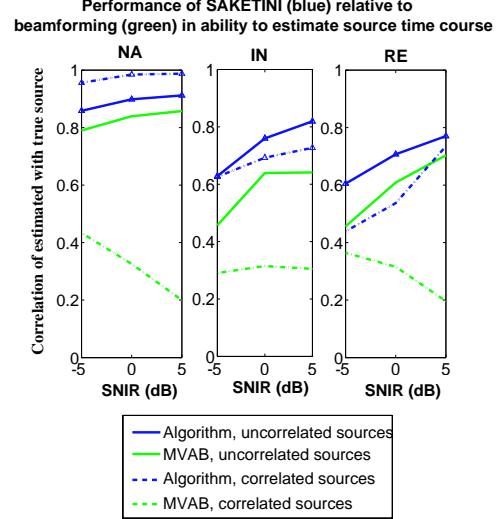

Figure 4: Performance of algorithm relative to beamforming for simulated datasets. See text for details.

100 trials collected while a human subject was alert but not performing tasks or receiving stimuli. This real background data thus includes real sensor noise plus real "ongoing" brain activity that could interfere with evoked sources and adds spatial correlation to the sensor data. We varied the Signal-to-Noise Ratio (SNR) and the corresponding Signal to Noise-plus-Interference Ratio (SNIR). SNIR is calculated from the ratio of the sensor data resulting from sources only to sensor data from noise plus interference. The first performance figure (Fig. 2) shows the localization error of the proposed method relative to the MVAB. For this data, a single dipole was placed randomly within

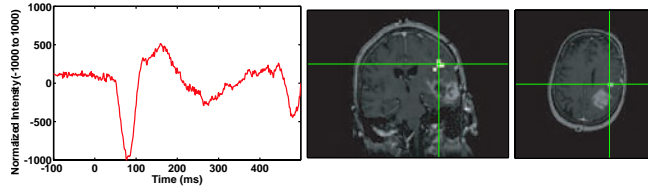

Figure 5: Algorithm applied to auditory MEG dataset in patient with temporal lobe tumor.

the voxel grid space. The largest peak in the likelihood map was found and the distance from this point to the true source was recorded. Each datapoint is an average of 20 realizations of the source configuration, with error bars showing standard error. This simulation was performed for a variety of SNIR's and for all three cases of noise described above. The results from NA were omitted since both the proposed method and MVAB performed perfectly (zero error). This figure clearly shows that the error in localization is smaller for the proposed method (black) than for MVAB (green).

The next set of simulations examines the proposed method's ability to estimate the source time course $s_n$. Three sources were placed in the brain voxel grid. The locations of these sources were fixed, but the orientation and time courses were allowed to vary across realizations of the simulations. In half the cases, two of the three sources were forced to be perfectly correlated in time (a scenario where the MVAB is known to fail), while the time course of the third source was random relative to the other two. An example of the likelihood map and estimated time courses are shown in Fig. 3. The likelihood map from the proposed method (on the left) has peaks near all three sources, including the two that were perfectly correlated (depicted by squares). However, the MVAB (middle plot) largely misses the source on the left. On the right plot, the estimated time courses from the proposed method (dashes) and MVAB (dots) are plotted relative to the true time course (solid). The top and middle plots correspond to the (square) correlated sources. While both methods estimate the time courses well, MVAB underestimates the overall strength of the source on the top plot, and exhibits extra noise in the pre-stimulus period for the middle plot.

The performance of the proposed model on the same set of simulations of correlated sources, compared to beamforming, are shown in Fig. 4. This figure shows the correlation of the estimated with the true time course, for three cases of NA, IN, and RE, and for both correlated and uncorrelated sources, as a function of SNIR. The proposed method consistently out-performs the MVAB whether the simulated sources are highly correlated with each other (dashed lines), or uncorrelated (solid), and especially in the RE case. Each datapoint represents an average of 10 realizations of the simulation, with standard errors on the order of 0.05 (not shown).

## 3.2   Real data

Stimulus-evoked data was collected in a 275-channel CTF System MEG device from a patient with temporal lobe tumor near auditory cortex. The stimulus was a noise burst presented binaurally in 120 trials. A large peak is typically seen around 100ms after presentation of an auditory stimulus, termed the M100 peak. Figure 5 shows the results of the proposed method applied to this dataset. On the right, the likelihood map show a spatial peak in auditory cortex near the tumor. At that peak voxel, the time course was extracted and plotted on the left, showing the clear M100 peak. This information can be useful to the neurosurgical team for guiding the location of surgical lesion and for providing knowledge of the patient's auditory processing abilities.

We next tested the proposed method on its ability to localize interictal spikes obtained from a patient with epilepsy. No sensory stimuli were presented to this patient in this dataset, which was collected in the same MEG device described above. A Registered EEG/Evoked Potential Technologist marked segments of the continuously-collected dataset which contained spontaneous spikes, as well as segments that clearly contained no spikes. One segment of data with a spike marked at 400ms was used here as the "post-stimulus" period and a separate, spike-free, segment of equal length was used as the "pre-stimulus" period. Figure 6 shows the proposed method's performance on this dataset. The top left subplot shows the raw sensor data for the segment containing the marked spike. The bottom left shows the location of the equivalent-current dipole (ECD) fit to several spikes from this patient; this location from the ECD fit would normally be used clinically. The middle bottom figure shows the likelihood map from the proposed model; the peak is in clear agreement with the standard ECD localization. The middle top figure shows the time course estimated for the likelihood spatial peak.

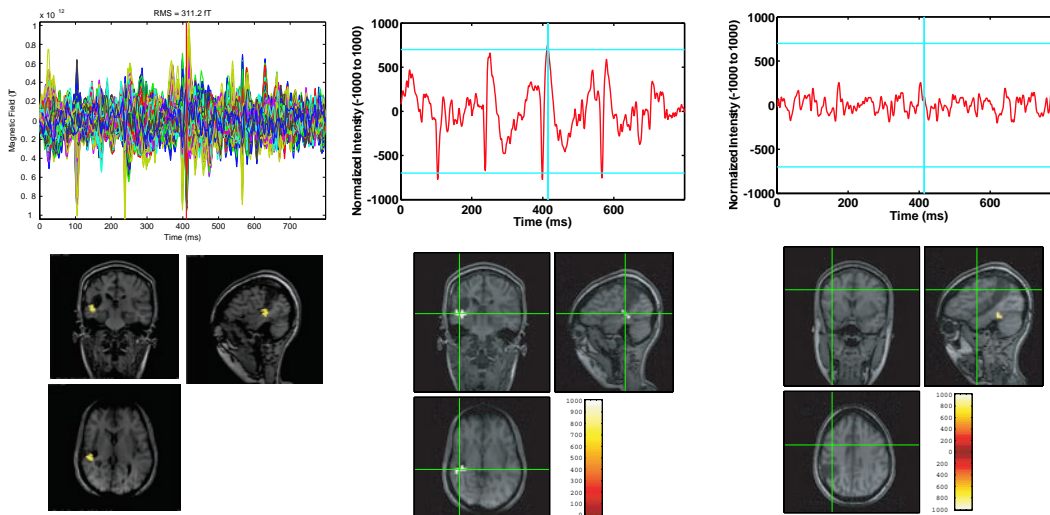

Figure 6: Performance of algorithm applied to data from an epileptic patient. See text for details.

The spike at 400ms is clearly seen; this cleaned waveform could be of use to the clinician in analyzing peak shape. Finally, the top right plot shows a source time course from a randomly selected location far from the epileptic spike source (shown with cross-hairs on bottom right plot), in order to show the low noise level and to show lack of cross-talk onto source estimates elsewhere.

## 4   Extensions

We have described a novel probabilistic algorithm which performs source localization while robust to interference and demonstrated its superior performance over a standard method in a variety of simulations and real datasets. The model takes advantage of knowledge of when sources of interest are not occurring (such as in the pre-stimulus period of a evoked response paradigm). This model currently assumes averaged data from an evoked response paradigm, but could be extended to examine variations from the average in individual trials, only involving a few extra parameters to estimate. Furthermore, the model could be extended to take advantage of temporal smoothness in the data as well as frequency content. Additionally, spatial smoothness or spatial priors from other modalities, such as structural or functional MRI, could be incorporated. Furthermore, one is not limited to $s_n$ in a single voxel; the above formulation holds for any $P$ arbitrarily chosen dipole components, no matter which voxels they belong to, and for any value of $P$. Of course, as $P$ increases the inferred value of $\Phi$ becomes less accurate, and one might choose to restrict it to a diagonal or block-diagonal form.

## References

[1] K. Sekihara, M. Sahani, and S.S. Nagarajan, "Localization bias and spatial resolution of adaptive and non-adaptive spatial filters for MEG source reconstruction," *NeuroImage*, vol. 25, pp. 1056–1067, 2005.

[2] K. Sekihara, S.S. Nagarajan, D. Poeppel, and A. Marantz, "Performance of an MEG adaptive-beamformer technique in the presence of correlated neural activities: Effects on signal intensity and time-course estimates," *IEEE Trans Biomed Eng*, vol. 49, pp. 1534–1546, 2002.

[3] Srikantan S. Nagarajan, Hagai T. Attias, Kenneth E. Hild, and Kensuke Sekihara, "A graphical model for estimating stimulus-evoked brain responses from magnetoencephalography data with large background brain activity," *Neuroimage*, vol. 30, pp. 400–416, 2006.

[4] S.S. Nagarajan, H.T. Attias, K.E. Hild, and K. Sekihara, "Stimulus evoked independent factor analysis of MEG data with large background activity," in *Adv. Neur. Info. Proc. Sys.*, 2005.

[5] J. Sarvas, "Basic mathematical and electromagnetic concepts of the biomagnetic inverse problem," *Phys Med Biol*, vol. 32, pp. 11–22, 1987.
